# Maximum Likelihood Blind Source Separation: A Context-Sensitive Generalization of ICA

**Barak A. Pearlmutter**
Computer Science Dept, FEC 313
University of New Mexico
Albuquerque, NM 87131
bap@cs.unm.edu

**Lucas C. Parra**
Siemens Corporate Research
755 College Road East
Princeton, NJ 08540-6632
lucas@scr.siemens.com

## Abstract

In the square linear blind source separation problem, one must find a linear unmixing operator which can detangle the result $x_i(t)$ of mixing $n$ unknown independent sources $s_i(t)$ through an unknown $n \times n$ mixing matrix $\mathbf{A}(t)$ of causal linear filters: $x_i = \sum_j a_{ij} * s_j$. We cast the problem as one of maximum likelihood density estimation, and in that framework introduce an algorithm that searches for independent components using both temporal and spatial cues. We call the resulting algorithm "Contextual ICA," after the (Bell and Sejnowski 1995) Infomax algorithm, which we show to be a special case of cICA. Because cICA can make use of the temporal structure of its input, it is able separate in a number of situations where standard methods cannot, including sources with low kurtosis, colored Gaussian sources, and sources which have Gaussian histograms.

## 1 The Blind Source Separation Problem

Consider a set of $n$ indepent sources $s_1(t), \ldots, s_n(t)$. We are given $n$ linearly distorted sensor reading which combine these sources, $x_i = \sum_j a_{ij} s_j$, where $a_{ij}$ is a filter between source $j$ and sensor $i$, as shown in figure 1a. This can be expressed as

$$x_i(t) = \sum_j \sum_{\tau=0}^{\infty} a_{ji}(\tau) s_j(t - \tau) = \sum_j a_{ji} * s_j$$

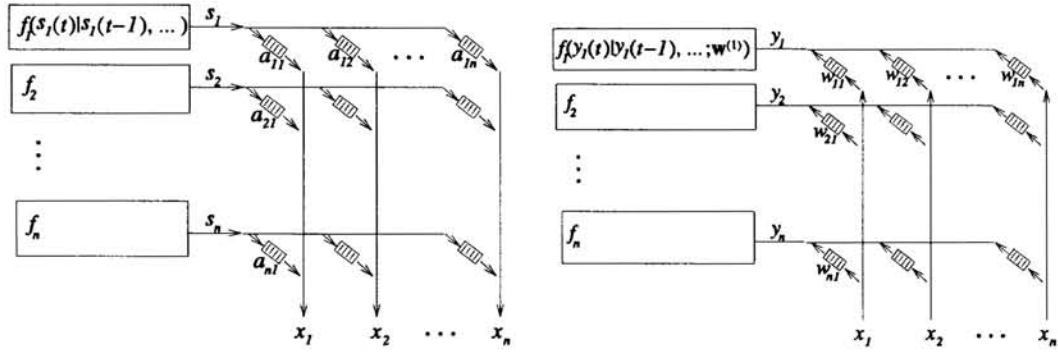

Figure 1: The left diagram shows a generative model of data production for blind source separation problem. The cICA algorithm fits the reparametrized generative model on the right to the data. Since (unless the mixing process is singular) both diagrams give linear maps between the sources and the sensors, they are mathematically equivalent. However, (a) makes the transformation from **s** to **x** explicit, while (b) makes the transformation from **x** to **y**, the estimated sources, explicit.

or, in matrix notation, $\mathbf{x}(t) = \sum_{\tau=0}^{\infty} A(\tau)\mathbf{s}(t - \tau) = A * \mathbf{s}$. The square linear blind source separation problem is to recover $s$ from $x$. There is an inherent ambiguity in this, for if we define a new set of sources $s'$ by $s_i' = b_i * s_i$ where $b_i(\tau)$ is some invertable filter, then the various $s_i'$ are independent, and constitute just as good a solution to the problem as the true $s_i$, since $x_i = \sum_j (a_{ij} * b_j^{-1}) * s_j'$. Similarly the sources could be arbitrarily permuted.

Surprisingly, up to permutation of the sources and linear filtering of the individual sources, the problem is well posed—assuming that the sources $s_j$ are *not* Gaussian. The reason for this is that only with a correct separation are the recovered sources truly statistically independent, and this fact serves as a sufficient constraint. Under the assumptions we have made,[1] and further assuming that the linear transformation $A$ is invertible, we will speak of recovering $y_i(t) = \sum_j w_{ji} * x_j$ where these $y_i$ are a filtered and permuted version of the original unknown $s_i$. For clarity of exposition, will often refer to "the" solution and refer to the $y_i$ as "the" recovered sources, rather than refering to an point in the manifold of solutions and a set of consistent recovered sources.

## 2   Maximum likelihood density estimation

Following Pham, Garrat, and Jutten (1992) and Belouchrani and Cardoso (1995), we cast the BSS problem as one of maximum likelihood density estimation. In the MLE framework, one begins with a probabilistic model of the data production process. This probabilistic model is parametrized by a vector of modifiable parameters **w**, and it therefore assigns a **w**-dependent probability density $p(\mathbf{x}_0, \mathbf{x}_1, \ldots; \mathbf{w})$ to a each possible dataset $\mathbf{x}_0, \mathbf{x}_1, \ldots$. The task is then to find a **w** which maximizes this probability.

There are a number of approaches to performing this maximization. Here we apply

the stochastic gradient method, in which a single stochastic sample $\mathbf{x}$ is chosen from the dataset and $-dlogp(\mathbf{x};\mathbf{w})/d\mathbf{w}$ is used as a stochastic estimate of the gradient of the negative likelihood $\sum_t -dlogp(\mathbf{x}(t);\mathbf{w})/d\mathbf{w}$.

## 2.1 The likelihood of the data

The model of data production we consider is shown in figure 1a. In that model, the sensor readings $\mathbf{x}$ are an explicit linear function of the underlying sources $\mathbf{s}$.

In this model of the data production, there are two stages. In the first stage, the sources independently produce signals. These signals are time-dependent, and the probability density of source $i$ producing value $s_j(t)$ at time $t$ is $f_j(s_j(t)|s_j(t-1), s_j(t-2), \ldots)$. Although this source model could be of almost any differentiable form, we used a generalized autoregressive model described in appendix A. For expository purposes, we can consider using a simple AR model, so we model $s_j(t) = b_j(1)s_j(t-1) + b_j(2)s_j(t-2) + \cdots + b_j(T)s_j(t-T) + r_j$, where $r_j$ is an iid random variable, perhaps with a complicated density.

It is important to distinguish two different, although related, linear filters. When the source models are simple AR models, there are two types of linear convolutions being performed. The first is in the way each source produces its signal: as a linear function of its recent history plus a white driving term, which could be expressed as a moving average model, a convolution with a white driving term, $s_j = b'_j * r_j$. The second is in the way the sources are mixed: linear functions of the output of each source are added, $x_i = \sum_j a_{ji} * s_j = \sum_j (a_{ji} * b'_j) * r_j$. Thus, with AR sources, the source convolution could be folded into the convolutions of the linear mixing process.

If we were to estimate values for the free parameters of this model, *i.e.* to estimate the filters, then the task of recovering the estimated sources from the sensor output would require inverting the linear $A = (a_{ij})$, as well as some technique to guarantee its non-singularity. Such a model is shown in figure 1a. Instead, we parameterize the model by $W = A^{-1}$, an estimated unmixing matrix, as shown in figure 1b. In this indirect representation, $\mathbf{s}$ is an explicit linear function of $\mathbf{x}$, and therefore $\mathbf{x}$ is only an *implicit* linear function of $\mathbf{s}$. This parameterization of the model is equally convenient for assigning probabilities to samples $\mathbf{x}$, and is therefore suitable for MLE. Its advantage is that because the transformation from sensors to sources is estimated explicitly, the sources can be recovered directly from the data and the estimated model, without inversion. Note that in this inverse parameterization, the estimated mixture process is stored in inverse form. The source-specific models $f_i$ are kept in forward form. Each source-specific model $i$ has a vector of parameters, which we denote $\mathbf{w}^{(i)}$.

We are now in a position to calculate the likelihood of the data. For simplicity we use a matrix $W$ of real numbers rather than FIR filters. Generalizing this derivation to a matrix of filters is straightforward, following the same techniques used by Lambert (1996), Torkkola (1996), A. Bell (1997), but space precludes a derivation here.

The individual generative source models give

$$p(\mathbf{y}(t)|\mathbf{y}(t-1), \mathbf{y}(t-2), \ldots) = \prod_i f_i(y_i(t)|y_i(t-1), y_i(t-2), \ldots) \qquad (1)$$

where the probability densities $f_i$ are each parameterized by vectors $\mathbf{w}^{(i)}$. Using these equations, we would like to express the likelihood of $\mathbf{x}(t)$ in closed form, given the history $\mathbf{x}(t-1), \mathbf{x}(t-2), \ldots$. Since the history is known, we therefore also know the history of the recovered sources, $\mathbf{y}(t-1), \mathbf{y}(t-2), \ldots$. This means that we can calculate the density $p(\mathbf{y}(t)|\mathbf{y}(t-1), \ldots)$. Using this, we can express the density of $\mathbf{x}(t)$ and expand $\widehat{G} = \log \hat{p}(\mathbf{x}; \mathbf{w}) = \log|\mathbf{W}| + \sum_j \log f_j(y_j(t)|y_j(t-1), y_j(t-2), \ldots; \mathbf{w}^{(j)})$ There are two sorts of parameters which we must take the derivative with respect to: the matrix $W$ and the source parameters $\mathbf{w}^{(j)}$. The source parameters do not influence our recovered sources, and therefore have a simple form

$$\frac{d\widehat{G}}{d\mathbf{w}_j} = -\frac{df_j(y_j; \mathbf{w}_j)/d\mathbf{w}_j}{f_j(y_j; \mathbf{w}_j)}$$

However, a change to the matrix $W$ changes $\mathbf{y}$, which introduces a few extra terms. Note that $d\log|W|/dW = W^{-T}$, the transpose inverse. Next, since $\mathbf{y} = W\mathbf{x}$, we see that $dy_j/dW = (0|\mathbf{x}|0)^T$, a matrix of zeros except for the vector $\mathbf{x}$ in row $j$. Now we note that $df_j(\cdot)/dW$ term has two logical components: the first from the effect of changing $W$ upon $y_j(t)$, and the second from the effect of changing $W$ upon $y_j(t-1), y_j(t-2), \ldots$. (This second is called the "recurrent term", and such terms are frequently dropped for convenience. As shown in figure 3, dropping this term here is not a reasonable approximation.)

$$\frac{df_j(y_j(t)|y_j(t-1), \ldots; \mathbf{w}_j)}{dW} = \frac{\partial f_j}{\partial y_j(t)} \frac{dy_j(t)}{dW} + \sum_\tau \frac{\partial f_j}{\partial y_j(t-\tau)} \frac{dy_j(t-\tau)}{dW}$$

Note that the expression $\frac{dy_j(t-\tau)}{dW}$ is the only matrix, and it is zero except for the $j$th row, which is $\mathbf{x}(t-\tau)$. The expression $\partial f_j/\partial y_j(t)$ we shall denote $f_j'(\cdot)$, and the expression $\partial f_j \partial y_j(t-\tau)$ we shall denote $f^{(\tau)}(\cdot)$. We then have

$$\frac{d\widehat{G}}{d\mathbf{W}} = -\mathbf{W}^{-T} - \left(\frac{f_j'(\cdot)}{f_j(\cdot)}\right)_j \mathbf{x}(t)^T - \sum_{\tau=1}^{\infty} \left(\frac{f_j^{(\tau)}(\cdot)}{f_j(\cdot)}\right)_j \mathbf{x}(t-\tau)^T \qquad (2)$$

where $(expr(j))_j$ denotes the column vector whose elements are $expr(1), \ldots, expr(n)$.

## 2.2 The natural gradient

Following Amari, Cichocki, and Yang (1996), we follow a pseudogradient. Instead of using equation 2, we post-multiply this quantity by $W^T W$. Since this is a positive-definite matrix, it does not affect the stochastic gradient convergence criteria, and the resulting quantity simplifies in a fashion that neatly eliminates the costly matrix inversion otherwise required. Convergence is also accelerated.

## 3 Experiments

We conducted a number of experiments to test the efficacy of the cICA algorithm. The first, shown in figure 2, was a toy problem involving a set of processed deliberately constructed to be difficult for conventional source separation algorithms. In the second experiment, shown in figure 3, ten real sources were digitally mixed with an instantaneous matrix and separation performance was measured as a funciton of varying model complexity parameters. These sources have are available for benchmarking purposes in http://www.cs.unm.edu/~bap/demos.html.

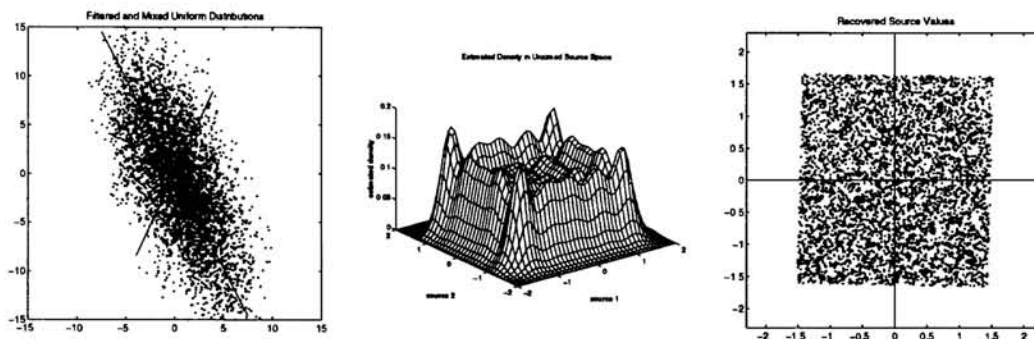

Figure 2: cICA using a history of one time step and a mixture of five logistic densities for each source was applied to 5,000 samples of a mixture of two one-dimensional uniform distributions each filtered by convolution with a decaying exponential of time constant of 99.5. Shown is a scatterplot of the data input to the algorithm, along with the true source axes (left), the estimated residual probability density (center), and a scatterplot of the residuals of the data transformed into the estimated source space coordinates (right). The product of the true mixing matrix and the estimated unmixing matrix deviates from a scaling and permutation matrix by about 3%.

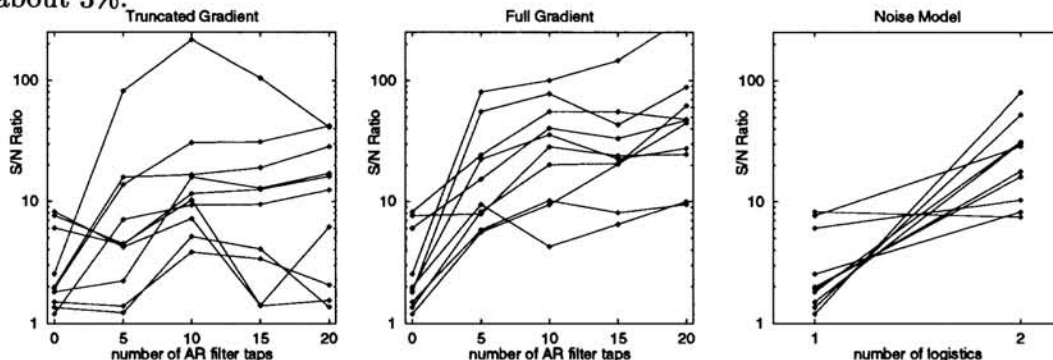

Figure 3: The performance of cICA as a function of model complexity and gradient accuracy. In all simulations, ten five-second clips taken digitally from ten audio CD were digitally mixed through a random ten-by-ten instantanious mixing matrix. The signal to noise ratio of each original source as expressed in the recovered sources is plotted. In (a) and (b), AR source models with a logistic noise term were used, and the number of taps of the AR model was varied. (This reduces to Bell-Sejnowski infomax when the number of taps is zero.) Is (a), the recurrent term of the gradient was left out, while in (b) the recurrent term was included. Clearly the recurrent term is important. In (c), a degenerate AR model with zero taps was used, but the noise term was a mixture of logistics, and the number of logistics was varied.

## 4 Discussion

The Infomax algorithm (Baram and Roth 1994) used for source separation (Bell and Sejnowski 1995) is a special case of the above algorithm in which (a) the mixing is not convolutional, so $W(1) = W(2) = \ldots = 0$, and (b) the sources are assumed to be iid, and therefore the distributions $f_i(y(t))$ are not history sensitive. Further, the form of the $f_i$ is restricted to a very special distribution: the logistic density,

the derivative of the sigmoidal function $1/(1 + \exp -\xi)$. Although ICA has enjoyed a variety of applications (Makeig *et al.* 1996; Bell and Sejnowski 1996b; Baram and Roth 1995; Bell and Sejnowski 1996a), there are a number of sources which it cannot separate. These include all sources with Gaussian histograms (*e.g.* colored gaussian sources, or even speech to run through the right sort of slight nonlinearity), and sources with low kurtosis. As shown in the experiments above, these are of more than theoretical interest.

If we simplify our model to use ordinary AR models for the sources, with gaussian noise terms of fixed variance, it is possible to derive a closed-form expression for $W$ (Hagai Attias, personal communication). It may be that for many sources of practical interest, trading away this model accuracy for speed will be fruitful.

## 4.1   Weakened assumptions

It seems clear that, in general, separating when there are fewer microphones than sources requires a strong bayesian prior, and even given perfect knowledge of the mixture process and perfect source models, inverting the mixing process will be computationally burdensome. However, when there are more microphones than sources, there is an opportunity to improve the performance of the system in the presence of noise. This seems straightforward to integrate into our framework. Similarly, fast-timescale microphone nonlinearities are easily incorporated into this maximum likelihood approach.

The structure of this problem would seem to lend itself to EM. Certainly the individual source models can be easily optimized using EM, assuming that they themselves are of suitable form.

## Footnotes

[1]Without these assumptions, for instance in the presence of noise, even a linear mixing process leads to an optimal unmixing process that is highly nonlinear.

## References

A. Bell, T.-W. L. (1997). Blind separation of delayed and convolved sources. In *Advances in Neural Information Processing Systems 9*. MIT Press. In this volume.

Amari, S., Cichocki, A., and Yang, H. H. (1996). A new learning algorithm for blind signal separation. In *Advances in Neural Information Processing Systems 8*. MIT Press.

Baram, Y. and Roth, Z. (1994). Density Shaping by Neural Networks with Application to Classification, Estimation and Forecasting. Tech. rep. CIS-94-20, Center for Intelligent Systems, Technion, Israel Institute for Technology, Haifa.

Baram, Y. and Roth, Z. (1995). Forecasting by Density Shaping Using Neural Networks. In *Computational Intelligence for Financial Engineering* New York City. IEEE Press.

Bell, A. J. and Sejnowski, T. J. (1995). An Information-Maximization Approach to Blind Separation and Blind Deconvolution. *Neural Computation*, *7*(6), 1129–1159.

Bell, A. J. and Sejnowski, T. J. (1996a). The Independent Components of Natural Scenes. *Vision Research*. Submitted.

Bell, A. J. and Sejnowski, T. J. (1996b). Learning the higher-order structure of a natural sound. *Network: Computation in Neural Systems*. In press.

Belouchrani, A. and Cardoso, J.-F. (1995). Maximum likelihood source separation by the expectation-maximization technique: Deterministic and stochastic implementation. In *Proceedings of 1995 International Symposium on Non-Linear Theory and Applications*, pp. 49–53 Las Vegas, NV. In press.

Lambert, R. H. (1996). *Multichannel Blind Deconvolution: FIR Matrix Algebra and Separation of Multipath Mixtures*. Ph.D. thesis, USC.

Makeig, S., Anllo-Vento, L., Jung, T.-P., Bell, A. J., Sejnowski, T. J., and Hillyard, S. A. (1996). Independent component analysis of event-related potentials during selective attention. *Society for Neuroscience Abstracts*, *22*.

Pearlmutter, B. A. and Parra, L. C. (1996). A Context-Sensitive Generalization of ICA. In *International Conference on Neural Information Processing* Hong Kong. Springer-Verlag. Url ftp://ftp.cnl.salk.edu/pub/bap/iconip-96-cica.ps.gz.

Pham, D., Garrat, P., and Jutten, C. (1992). Separation of a mixture of independent sources through a maximum likelihood approach. In *European Signal Processing Conference*, pp. 771–774.

Torkkola, K. (1996). Blind separation of convolved sources based on information maximization. In *Neural Networks for Signal Processing VI* Kyoto, Japan. IEEE Press. In press.

## A    Fixed mixture AR models

The $f_j(u_j; \mathbf{w}_j)$ we used were a mixture AR processes driven by logistic noise terms, as in Pearlmutter and Parra (1996). Each source model was

$$f_j(u_j(t)|u_j(t-1), u_j(t-2), \ldots; \mathbf{w}_j) = \sum_k m_{jk} \, h((u_j(t) - \bar{u}_{jk})/\sigma_{jk})/\sigma_{jk} \qquad (3)$$

where $\sigma_{jk}$ is a scale parameter for logistic density $k$ of source $j$ and is an element of $\mathbf{w}_j$, and the mixing coefficients $m_{jk}$ are elements of $\mathbf{w}_j$ and are constrained by $\sum_k m_{jk} = 1$. The component means $\bar{u}_{jk}$ are taken to be linear functions of the recent values of that source,

$$\bar{u}_{jk} = \sum_{\tau=1} a_{jk}(\tau) \, u_j(t - \tau) + b_{jk} \qquad (4)$$

where the linear prediction coefficients $a_{jk}(\tau)$ and bias $b_{jk}$ are elements of $\mathbf{w}_j$. The derivatives of these are straightforward; see Pearlmutter and Parra (1996) for details. One complication is to note that, after each weight update, the mixing coefficients must be normalized, $m_{jk} \leftarrow m_{jk}/\sum_{k'} m_{jk'}$.